# The effect of eligibility traces on finding optimal memoryless policies in partially observable Markov decision processes

**John Loch**
Department of Computer Science
University of Colorado
Boulder, CO 80309-0430
loch@cs.colorado.edu

## Abstract

Agents acting in the real world are confronted with the problem of making good decisions with limited knowledge of the environment. Partially observable Markov decision processes (POMDPs) model decision problems in which an agent tries to maximize its reward in the face of limited sensor feedback. Recent work has shown empirically that a reinforcement learning (RL) algorithm called Sarsa($\lambda$) can efficiently find optimal memoryless policies, which map current observations to actions, for POMDP problems (Loch and Singh 1998). The Sarsa($\lambda$) algorithm uses a form of short-term memory called an eligibility trace, which distributes temporally delayed rewards to observation-action pairs which lead up to the reward. This paper explores the effect of eligibility traces on the ability of the Sarsa($\lambda$) algorithm to find optimal memoryless policies. A variant of Sarsa($\lambda$) called k-step truncated Sarsa($\lambda$) is applied to four test problems taken from the recent work of Littman, Littman, Cassandra and Kaelbling, Parr and Russell, and Chrisman. The empirical results show that eligibility traces can be significantly truncated without affecting the ability of Sarsa($\lambda$) to find optimal memoryless policies for POMDPs.

## 1 Introduction

Agents which operate in the real world, such as mobile robots, must use sensors which at best give only partial information about the state of the environment. Information about the robot's surroundings is necessarily incomplete due to noisy and/or imperfect sensors, occluded objects, and the inability of the robot to know precisely where it is. Such agent-environment systems can be modeled as partially observable Markov decision processes or POMDPs (Sondik, 1978).

A variety of algorithms have been developed for solving POMDPs (Lovejoy, 1991). However most of these techniques do not scale well to problems involving more than a few dozen states due to the computational complexity of the solution methods (Cassandra, 1994; Littman 1994). Therefore, finding efficient reinforcement learning

methods for solving POMDPs is of great practical interest to the Artificial Intelligence and engineering fields.

Recent work has shown empirically that the Sarsa($\lambda$) algorithm can efficiently find the best deterministic memoryless policy for several POMDPs problems from the recent literature (Loch and Singh 1998). The empirical results from Loch and Singh (1998) suggest that eligibility traces are necessary for finding the best or optimal memoryless policy. For this reason, a variant of Sarsa($\lambda$) called k-step truncated Sarsa($\lambda$) is formulated to explore the effect of eligibility traces on the ability of Sarsa($\lambda$) to find the best memoryless policy.

The main contribution of this paper is to show empirically that a variant of Sarsa($\lambda$) using truncated eligibility traces can find the optimal memoryless policy for several POMDP problems from the literature. Specifically we show that the k-step truncated Sarsa($\lambda$) method can find the optimal memoryless policy for the four POMDP problems tested when $k \leq 2$.

## 2 Sarsa($\lambda$) and POMDPs

An environment is defined by a finite set of states S, the agent can choose from a finite set of actions A, and the agent's sensors provide it observations from a finite set X. On executing action a $\varepsilon$ A in state s $\varepsilon$ S the agent receives expected reward $r_s^a$ and the environment transitions to a state s' $\varepsilon$ S with probability $P^a_{ss'}$. The probability of the agent observing x $\varepsilon$ X given that the state is s is O(x|s).

A straightforward way to extend RL algorithms to POMDPs is to learn Q-value functions of observation-action pairs, i.e. to simply treat the agents observations as states. Below we describe the standard Sarsa($\lambda$) algorithm applied to POMDPs. At time step t the Q-value function is denoted $Q_t$ ; the eligibility trace function is denoted $\eta_t$ ; and the reward received is denoted $r_t$ . On experiencing transition $<x_t, a_t, r_t, x_{t+1}>$ the following updates are performed in order:

$$\eta_t(x_t, a_t) = 1$$

$$\eta_t(x, a) = \gamma\lambda \ \eta_{t-1}(x, a) \ ; \text{for all } x \neq x_t \text{ and } a \neq a_t$$

$$Q_{t+1}(x, a) = Q_t(x, a) + \alpha * \delta_t * \eta_t(x, a)$$

where $\delta_t = r_t + \gamma \ Q_t(x_{t+1}, a_{t+1}) - Q_t(x_t, a_t)$ and $\alpha$ is the step-size (learning rate). The eligibility traces are initialized to zero, and in episodic tasks they are reinitialized to zero after every episode. The greedy policy at time step t assigns to each observation x the action a = argmax$_b$ $Q_t$(x, b).

### 2.1 Sarsa($\lambda$) Using Truncated Eligibility Traces

Sarsa($\lambda$) with truncated eligibility traces uses a parameter k which sets the eligibility trace for an observation-action pair to zero if that observation-action pair was not visited within the last k-1 time steps. Thus 1-step truncated Sarsa($\lambda$) is equivalent to Sarsa(0) and 2-step truncated Sarsa($\lambda$) updates the Q-values of the current observation-action pair and the immediately preceding observation-action pair.

## 3 Empirical Results

The truncated Sarsa($\lambda$) algorithm was applied in an identical manner to four POMDP problems taken from the recent literature. Complete descriptions of the states, actions, observations, and rewards for each problem are provided in Loch and Singh (1998). Here we describe the aspects of the empirical results common to all four problems. At each step, the agent selected a random action with a probability equal to the exploration rate parameter and selected a greedy action otherwise. An initial exploration rate of 35% was used, decreasing linearly with each action (step) until the 350000th action from there onward the exploration rate remain fixed at 0%. Q-values were initialized to 0. Both the step-size $\alpha$ and the $\lambda$ values are held constant in each experiment. A discount factor $\gamma$ of 0.95 and a $\lambda$ value of 1.0 were used for all four problems.

### 3.1 Sutton's Grid World

Sutton's grid world (Littman 1994) is an agent-environment system with 46 states, 30 observations, and 4 actions. State transitions and observations are deterministic.

The 1-step truncated eligibility trace, equivalent to Sarsa(0), was able to find a policy which could only reach the goal from start states within 7 steps of the goal state as shown in Figure 1. The optimal memoryless policy yielding 416 total steps to the goal state was found by the 2-step, 4-step and 8-step truncated eligibility trace methods shown in Figure 1.

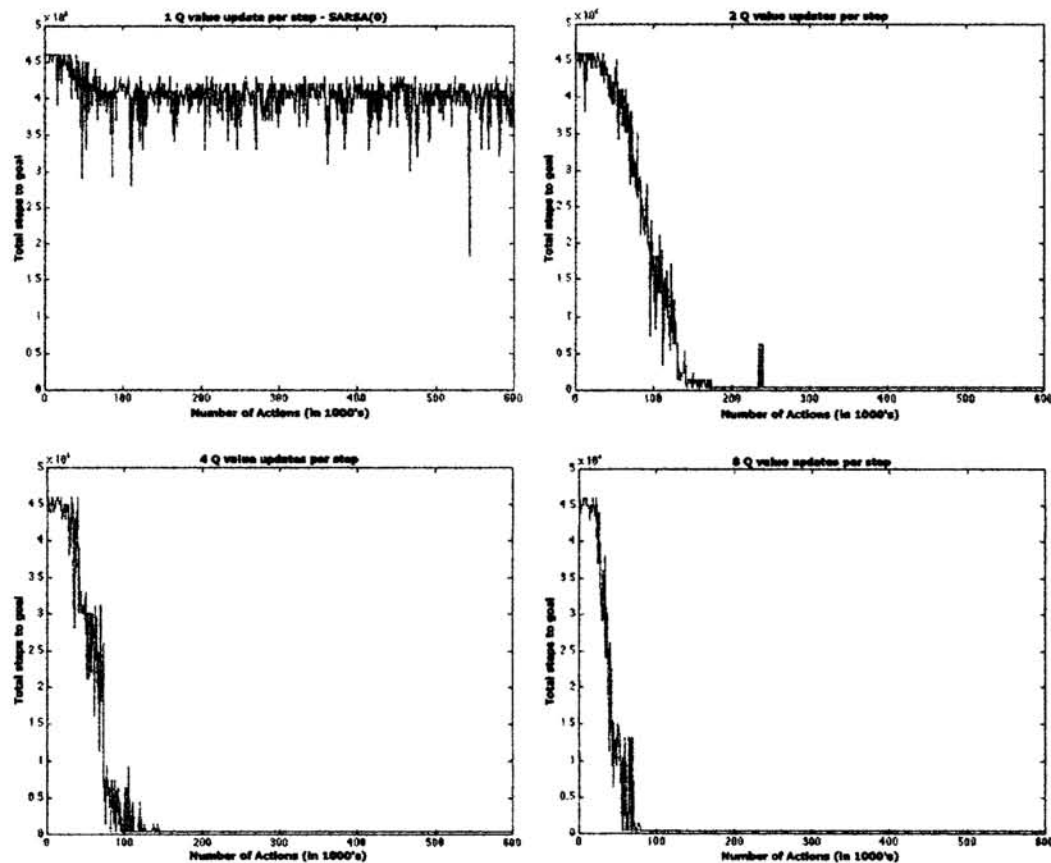

Figure 1: Sutton's Grid World (from Littman, 1994). Total steps to goal performance as a function of the number of learning steps for 1, 2, 4, and 8-step eligibility traces.

### 3.2 Chrisman's Shuttle Problem

Chrisman's shuttle problem is an agent-environment system with 8 states, 5 observations, and 3 actions. State transitions and observations are stochastic.

The 1-step truncated eligibility trace, equivalent to Sarsa(0), was unable to find a policy which could could reach the goal state (Figure 2). The optimal memoryless policy yielding an average reward per step of 1.02 was found by the 2-step, 4-step, and 8-step truncated eligibility trace methods shown in Figure 2.

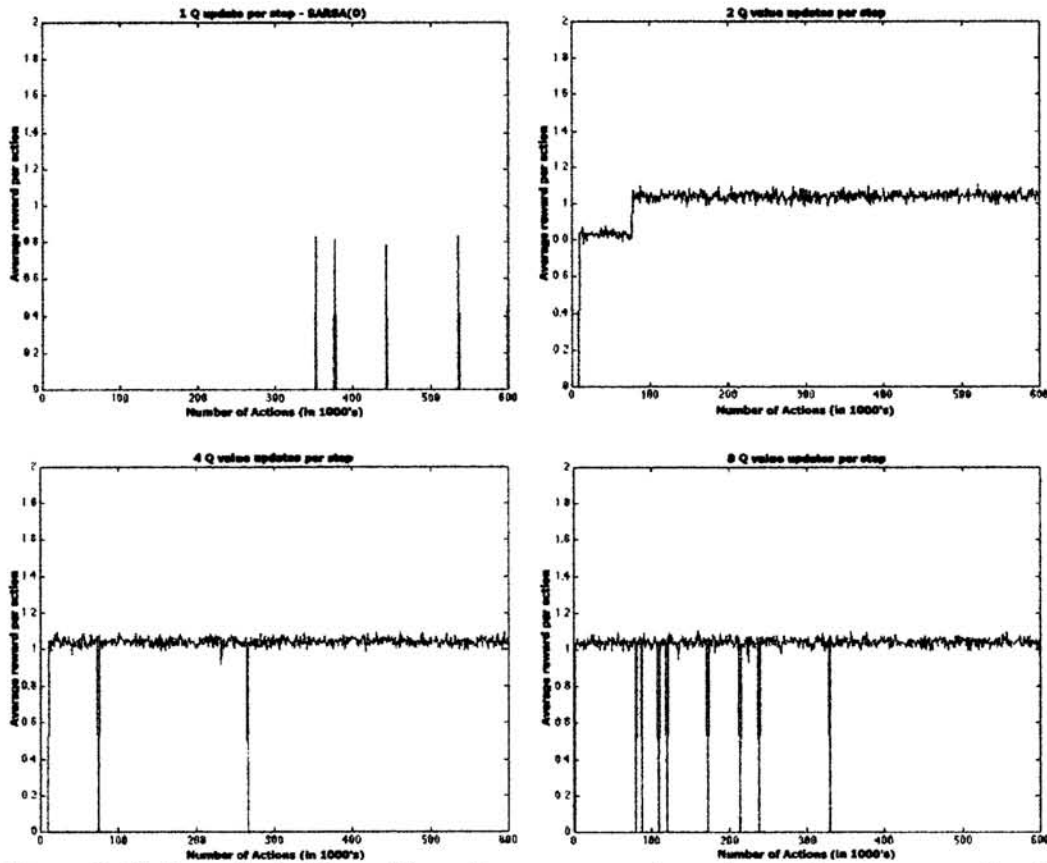

Figure 2: Chrisman's shuttle problem. Average reward per step performance as a function of the number of learning steps for 1, 2, 4, and 8-step eligibility traces.

### 3.3 Littman, Cassandra, and Kaelbling's 89 State Office World

Littman et al.'s 89 state office world (Littman 1995) is an agent-environment system with 89 states, 17 observations, and 5 actions. State transitions and observations are stochastic.

The 1-step truncated eligibility trace, equivalent to Sarsa(0), was able to find a policy which could reach the goal state in only 51% of the 251 trials (Figure 3). The 2-step, 4-step and 8-step truncated eligibility trace methods converged to the best memoryless policy found by Loch & Singh (1998) yielding a 77% success rate in reaching the goal state (Figure 3).

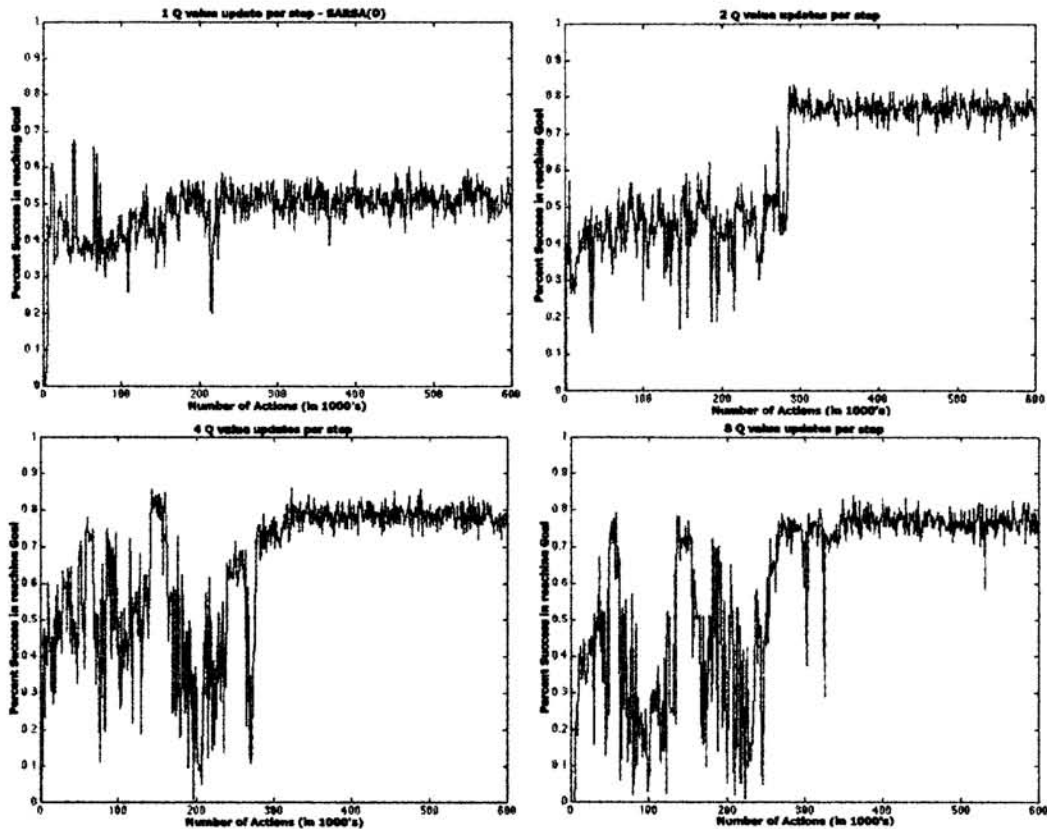

Figure 3: Littman et al.'s 89 state office world. Percent successful trials in reaching goal performance as a function of the number of learning steps for 1, 2, 4, and 8-step eligibility traces.

## 3.4 Parr & Russell's Grid World

Parr and Russell's grid world (Parr and Russell 1995) is an agent-environment system with 11 states, 6 observations, and 4 actions. State transitions are stochastic while observations are deterministic.

The optimal memoryless policy yielding an average reward per step of 0.024 was found by both the 1-step and 2-step truncated eligibility trace methods (Figure 4). Policies found by the 4-step and 8-step methods were not optimal. This result can be attributed to the sharp eligibility trace cutoff as this effect was not observed with smoothly decaying eligibility traces.

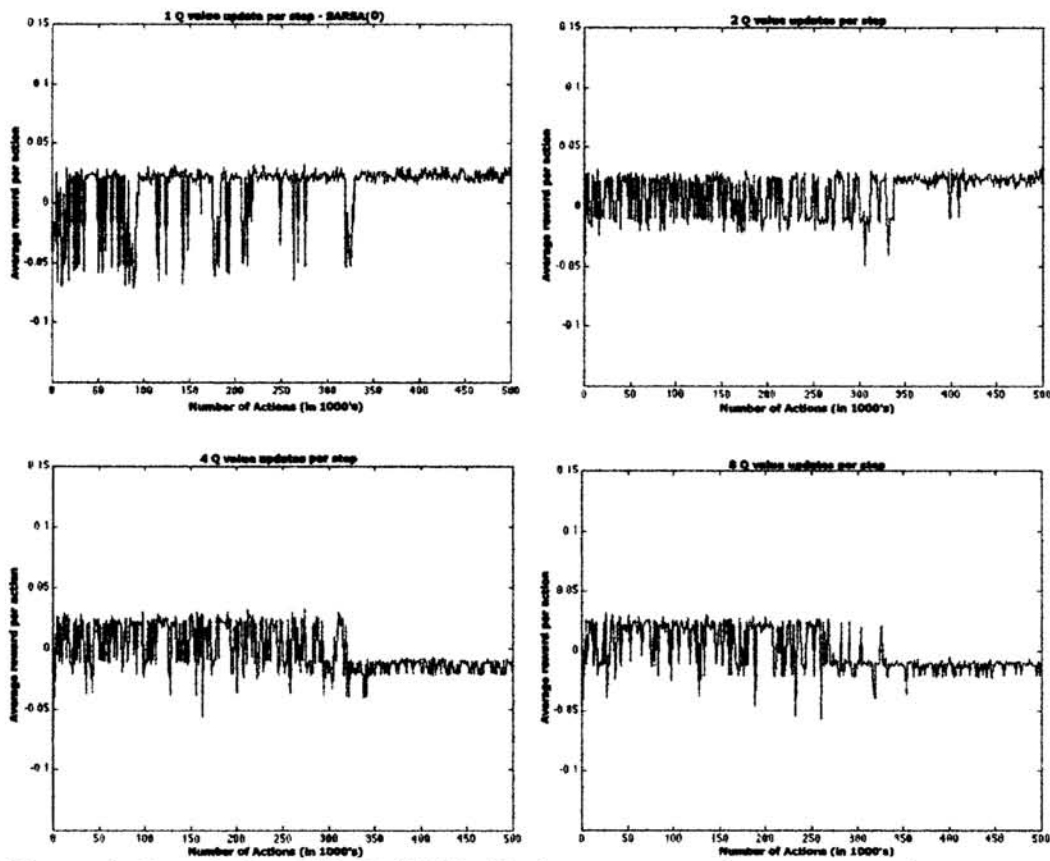

Figure 4: Parr & Russell's Grid World. Average reward per step performance as a function of the number of learning steps for 1, 2, 4, and 8-step eligibility traces.

### 3.5 Discussion

In all the empirical results presented above, we have shown that the k-step truncated Sarsa($\lambda$) algorithm was able to find the best or the optimal deterministic memoryless policy when k=2.

This result is surprising since it was expected that the length of the eligibility trace required to find a good or optimal policy would vary widely depending on problem specific factors such as landmark (unique observation) spacing and the delay between critical decisions and rewards. Several additional POMDP problems were formulated in an attempt to create a POMDP which would require a k value greater than 2 to find the optimal policy. However, for all trial POMDPs tested the optimal memoryless policy could be found with k $\leq$ 2.

## 4 Conclusions and Future Work

The ability of the Sarsa($\lambda$) algorithm and the k-step truncated Sarsa($\lambda$) algorithm to find optimal deterministic memoryless policies for a class of POMDP problems is important for several reasons. For POMDPs with good memoryless policies the Sarsa($\lambda$) algorithm provides an efficient method for finding the best policy in that space.

If the performance of the memoryless policy is unsatisfactory, the observation and action spaces of the agent can be modified so as to produce an agent with a good memoryless policy. The designer of the autonomous system or agent can modify the observation

space of the agent by either adding sensors or making finer distinctions in the current sensor values. In addition, the designer can add attributes from past observations into the current observation space. The action space can be modified by adding lower-level actions and by adding new actions to the space. Thus one method for designing a capable agent is to iterate between selecting an observation and action space for the agent, using Sarsa($\lambda$) to find the best memoryless policy in that space, and repeating until satisfactory performance is achieved.

This suggests a future line of research into how to automate the process of observation and action space selection so as to acheive an acceptable performance level. Other avenues of research include an exploration into theoretical reasons why Sarsa($\lambda$) and k-step truncated Sarsa($\lambda$) are able to solve POMDPs. In addition, further research needs to be conducted as to why short ($k \leq 2$) eligibility traces work well over a wide class of POMDPs.

# References

Cassandra, A. (1994). Optimal policies for partially observable Markov decision processes. Technical Report CS-94-14, Brown University, Department of Computer Science, Providence RI.

Littman, M. (1994). The Witness Algorithm: Solving partially observable Markov decision processes. Technical Report CS-94-40, Brown University, Department of Computer Science, Providence RI.

Littman, M., Cassandra, A., & Kaelbling, L. (1995). Learning policies for partially observable environments: Scaling up. In *Proceedings of the Twelfth International Conference on Machine Learning*, pages 362-370, San Francisco, CA, 1995. Morgan Kaufmann.

Loch, J., & Singh, S. (1998). Using eligibility traces to find the best memoryless policy in partially observable Markov decision processes. To appear In *Proceedings of the Fifteenth International Conference on Machine Learning,*, Madison, WI, 1998. Morgan Kaufmann. (Available from http://www.cs.colorado.edu/~baveja/papers.html)

Lovejoy, W. S. (1991). A survey of algorithmic methods for partially observable Markov decision processes. In *Annals of Operations Research*, 28:47-66.

Parr, R. & Russell, S. (1995). Approximating optimal policies for partially observable stochastic domains. In *Proceedings of the International Joint Conference on Artificial Intelligence*.

Sondik, E. J. (1978). The optimal control of partially observable Markov decision processes over the infinite horizon: Discounted costs. In *Operations Research*, 26(2).

Sutton, R.S. (1990). Integrated architectures for learning, planning, and reacting based on approximating dynamic programming. In *Proceedings of the Seventh International Conference of Machine Learning*, pages 216-224, San Mateo, CA. Morgan Kaufman.

Littman, M. (1994). Memoryless policies: theoretical limitations and practical results. In *From Animals to Animats 3: Proceedings of the Third International Conference on Simulation of Adaptive Behavior*, Cambridge, MA. MIT Press.
